# How biased are maximum entropy models?

**Jakob H. Macke**
Gatsby Computational Neuroscience Unit
University College London, UK
jakob@gatsby.ucl.ac.uk

**Iain Murray**
School of Informatics
University of Edinburgh, UK
i.murray@ed.ac.uk

**Peter E. Latham**
Gatsby Computational Neuroscience Unit
University College London, UK
pel@gatsby.ucl.ac.uk

## Abstract

Maximum entropy models have become popular statistical models in neuroscience and other areas in biology, and can be useful tools for obtaining estimates of mutual information in biological systems. However, maximum entropy models fit to small data sets can be subject to sampling bias; i.e. the true entropy of the data can be severely underestimated. Here we study the sampling properties of estimates of the entropy obtained from maximum entropy models. We show that if the data is generated by a distribution that lies in the model class, the bias is equal to the number of parameters divided by twice the number of observations. However, in practice, the true distribution is usually outside the model class, and we show here that this misspecification can lead to much larger bias. We provide a perturbative approximation of the maximally expected bias when the true model is out of model class, and we illustrate our results using numerical simulations of an Ising model; i.e. the second-order maximum entropy distribution on binary data.

## 1 Introduction

Over the last several decades, information theory [1, 2] has played a major role in our effort to understand the neural code in the brain [3, 4]. Its usefulness, however, is limited by the fact that the quantity of interest, mutual information (typically between stimuli and neuronal responses) is hard to compute from data [5]. Consequently, although this approach has led to a relatively deep understanding of neural coding in single neurons [4], it has told us far less about populations [6, 7]. In essence, the brute-force approaches to measuring mutual information that have worked so well on single spike trains simply do not work on populations. This is because the key-ingredient of mutual information is the entropy, and in general, estimation of the entropy from finite data sets suffers from a severe downward bias [8, 9]: on average, the entropy estimated on the data set will be lower than the actual entropy of the underlying model. While a number of improved estimators have been developed (see [5, 10] for an overview), the amount of data one needs is, ultimately, exponential in the number of neurons, so even modest populations (tens of neurons) are out of reach.

To apply information-theoretic techniques to populations, then, our only hope is to develop models in which the number of unconstrained parameters grows (relatively) slowly with the number of neurons [11]. For such models, estimating information requires much less data than brute force methods. Still, the amount of data is nontrivial, and naive estimators of information can be badly biased. Here we consider one class of models – maximum entropy models subject to linear constraints – and compute the bias in the entropy. We show that if the true distribution lies in the parametric model class, then the bias is equal to the number of parameters divided by twice the number of observations. When the true distribution is outside the model class, however, the bias can be much larger.

We illustrate our results using a very popular model in neuroscience, the Ising model [12], which is the second-order maximum entropy distribution on binary data. Recently, this model has become a popular means of characterizing the distribution of firing patterns in multi-electrode recordings, and has been used extensively in a wide range of applications, including recordings in the retina [13, 14] and visual cortex [15]. In addition, several recent studies [16, 17, 18] have used numerical simulations of large Ising models to understand the scaling of the entropy of the model with population size. And, finally, Ising models have been used in other fields in biology, for example to model gene-regulation networks [19].

## 2 Theory

### 2.1 Maximum entropy models

Our starting point is an underlying true distribution, denoted $p(\mathbf{x})$ where $\mathbf{x}$ is a (typically real valued) vector; the goal is to model it with a maximum entropy distribution. For simplicity, when developing the formalism we take $\mathbf{x}$ to be discrete; however, all our results apply to continuous variables.

The maximum entropy distribution is the distribution with the highest entropy subject to a set of constraints, where the entropy is given by

$$S = -\sum_{\mathbf{x}} p(\mathbf{x}) \log p(\mathbf{x}) . \tag{1}$$

Specifically, suppose that under the true distributions a set of $m$ functions, denoted $g_i(\mathbf{x})$, $i = 1, ..., m$, average to $\mu_i$,

$$\mu_i = \sum_{\mathbf{x}} p(\mathbf{x}) g_i(\mathbf{x}) . \tag{2}$$

If we use $q(\mathbf{x}|\boldsymbol{\mu})$ to denote the maximum entropy distribution (with $\boldsymbol{\mu} \equiv (\mu_1, \mu_2, ..., \mu_m)$), the constraints (here taken to be linear in the probability) are of the form

$$\sum_{\mathbf{x}} q(\mathbf{x}|\boldsymbol{\mu}) g_i(\mathbf{x}) = \mu_i . \tag{3}$$

Finding an explicit expression for $q(\mathbf{x}|\boldsymbol{\mu})$ is a straightforward optimization problem (see, e.g., [2]). It can be shown that the maximum entropy distribution is in the exponential family,

$$q(\mathbf{x}|\boldsymbol{\mu}) = \frac{\exp\left[\sum_{i=1}^{m} \lambda_i(\boldsymbol{\mu}) g_i(\mathbf{x})\right]}{Z(\boldsymbol{\mu})} \tag{4}$$

where the parameters, $\lambda_i$ (the Lagrange multipliers of the optimization problem), are chosen such that the constraints in Eq. (2) are satisfied. The partition function, $Z(\boldsymbol{\mu})$, ensures that the probabilities normalize to one,

$$Z(\boldsymbol{\mu}) = \sum_{\mathbf{x}} \exp\left[\sum_{i=1}^{m} \lambda_i(\boldsymbol{\mu}) g_i(\mathbf{x})\right] . \tag{5}$$

Once we have identified the parameters of this model, we can insert Eq. (4) into Eq. (1), which allows us to write the entropy in the form

$$S_q(\boldsymbol{\mu}) = \log Z(\boldsymbol{\mu}) - \sum_{i=1}^{m} \lambda_i(\boldsymbol{\mu}) \mu_i . \tag{6}$$

### 2.2 Estimation bias in maximum entropy models

So far we have assumed that the true $\mu_i$ are known. In general, though, we have to estimate the $\mu_i$ from data. Specifically, if we have $K$ observations of $\mathbf{x}$, denoted $\mathbf{x}^{(k)}$, $k = 1, ..., K$, then the estimate of $\mu_i$, denoted $\hat{\mu}_i$, is given by

$$\hat{\mu}_i = \frac{1}{K} \sum_{k=1}^{K} g_i\left(\mathbf{x}^{(k)}\right) . \tag{7}$$

We can still use the maximum entropy formulation described above; the only difference is that we replace $\boldsymbol{\mu}$ by $\hat{\boldsymbol{\mu}}$. Thus, the maximum entropy distribution is given by $q(\mathbf{x}|\hat{\boldsymbol{\mu}})$ (Eq. (4)) and the entropy by $S_q(\hat{\boldsymbol{\mu}})$ (Eq. (6)).

Because of sampling error, the $\hat{\mu}_i$ are not equal to their true values, $\mu_i$; consequently, neither is $S_q(\hat{\boldsymbol{\mu}})$. This leads to variability, in the sense that different sets of $\mathbf{x}^{(k)}$ lead to different entropies and, because the entropy is concave, to bias. Thus, the entropy estimated from a finite data set will be lower, on average, than the entropy obtained from the true underlying model. In the large $K$ limit, so that $\hat{\mu}_i$ is close to $\mu_i$, the bias can be computed by Taylor expanding around $S_q(\boldsymbol{\mu})$ and averaging over the true distribution, $p(\mathbf{x})$. Anticipating somewhat our result, we use $-b/2K$ to denote the bias, and we have

$$-\frac{b}{2K} \equiv \langle S_q(\hat{\boldsymbol{\mu}}) - S_q(\boldsymbol{\mu})\rangle_{p(\mathbf{x})} = \sum_{i=1}^{m} \frac{\partial S_q(\boldsymbol{\mu})}{\partial \mu_i}\langle\delta\mu_i\rangle_{p(\mathbf{x})} + \frac{1}{2}\sum_{i,j=1}^{m} \frac{\partial^2 S_q(\boldsymbol{\mu})}{\partial \mu_i \partial \mu_j}\langle\delta\mu_i\delta\mu_j\rangle_{p(\mathbf{x})} + ... \tag{8}$$

where

$$\delta\mu_i \equiv \hat{\mu}_i - \mu_i = \frac{1}{K}\sum_{k=1}^{K} g_i\left(\mathbf{x}^{(k)}\right) - \mu_i. \tag{9}$$

The angle brackets with subscript $p(\mathbf{x})$ indicate an average with respect to the true distribution, $p(\mathbf{x})$. The quantity we focus on is $b$, the normalized bias (as it is independent of $K$ in the large $K$ limit). Computing the averages and derivatives in Eq. (8) is straightforward (see Appendix A in the supplementary material for details), and we find that, through second order in $\delta\mu$,

$$b = \sum_{ij} C_{ij}^{q^{-1}} C_{ji}^{p}, \tag{10}$$

where

$$C_{ij}^{q} \equiv \langle\delta g_i(\mathbf{x})\delta g_j(\mathbf{x})\rangle_{q(\mathbf{x}|\boldsymbol{\mu})} \tag{11a}$$

$$C_{ij}^{p} \equiv \langle\delta g_i(\mathbf{x})\delta g_j(\mathbf{x})\rangle_{p(\mathbf{x})}. \tag{11b}$$

Here $C_{ij}^{q^{-1}}$ denotes the $ij^{\text{th}}$ entry of $C^{q-1}$ and

$$\delta g_i(\mathbf{x}) \equiv g_i(\mathbf{x}) - \mu_i. \tag{12}$$

### 2.3 Bias when the true model is in the model class

Equation (10) tells us the normalized bias (to first order in $1/K$). Evaluating it is, typically, hard, but there is one case in which we can write down an explicit expression for it: when the true distribution lies in the model class, so that $p(\mathbf{x}) = q(\mathbf{x}|\boldsymbol{\mu})$. In that case, $\mathbf{C}^q = \mathbf{C}^p$, the normalized bias is the trace of the identity matrix, and we have $b = m$ (recall that $m$ is the number of constraints); alternatively, $\text{Bias}[S] = -m/2K$.

An important within-model-class case arises when $\mathbf{x}$ is discrete and the "parametrized" model is a direct histogram of the data. If $\mathbf{x}$ can take on $D$ values, then there are $D - 1$ parameters (the "$-1$" comes from the fact that $p(\mathbf{x})$ must sum to 1) and the normalized bias is $(D - 1)/2K$. We thus recover a general version of the Miller–Madow [8] or Panzeri & Treves bias correction [9], which was derived for a multinomial distribution. (Note that our expression differs from theirs by a factor of $\log 2$; that's because they use base 2 logarithms whereas we use natural logarithms.) Alternatively, one can exploit the relationship between entropy-maximization and maximum-likelihood estimation in the exponential family to deduce this result from the asymptotic distribution of maximum likelihood estimators [20]. For details see Appendix B in the supplementary material.

### 2.4 Bias when the true model is not in the model class

In practice, it is rare for the true distribution to lie in the model class, so it is important to know how the normalized bias behaves in general. In this section, we investigate how quickly it changes when we leave the model class. We concentrate on the worst case scenario and determine the largest normalized bias that is consistent with a given "distance" from the true model class. For cases in which we are close to the true model class, we provide a perturbative expression for this quantity.

To assess the normalized bias out of model class, we assume that $p(x)$, the distribution from which the data was generated, can be written as

$$p(\mathbf{x}) = q(\mathbf{x}|\boldsymbol{\mu}) + \delta p(\mathbf{x}) \tag{13}$$

with $\delta p(\mathbf{x})$ chosen so that it is orthogonal to all the constraints; that is $\sum_{\mathbf{x}} \delta p(\mathbf{x}) g_i(\mathbf{x}) = 0$, which in turn implies that

$$\sum_{\mathbf{x}} p(\mathbf{x}) g_i(\mathbf{x}) = \sum_{\mathbf{x}} q(\mathbf{x}|\boldsymbol{\mu}) g_i(\mathbf{x}) \tag{14}$$

(and both, of course, are equal to $\mu_i$). We then ask how the normalized bias behaves as $\delta p(\mathbf{x})$ varies.

Because $q(\mathbf{x}|\boldsymbol{\mu})$ is independent of $\delta p(\mathbf{x})$, so is $C_{ij}^q$, and the normalized bias, $b$, that appears in Eq. (10) can be written (using Eq. (11b))

$$b = \langle B(\mathbf{x}) \rangle_{p(\mathbf{x})} \tag{15}$$

where

$$B(\mathbf{x}) \equiv \sum_{ij} \delta g_i(\mathbf{x}) C_{ij}^{q^{-1}} \delta g_j(\mathbf{x}) \,. \tag{16}$$

It's not possible to say anything definitive about the normalized bias in general, but what we can do is compute its maximum as a function of the distance between $p(\mathbf{x})$ and $q(\mathbf{x}|\boldsymbol{\mu})$, with "distance" measured by the Kullback–Leibler divergence. The latter quantity, denoted $\Delta S$, is given by

$$\Delta S = \sum_{\mathbf{x}} p(\mathbf{x}) \log \frac{p(\mathbf{x})}{q(\mathbf{x}|\boldsymbol{\mu})} = S_q(\boldsymbol{\mu}) - S_p \tag{17}$$

where $S_p$ is the entropy of $p(\mathbf{x})$. The second equality follows from the definition of $q(\mathbf{x}|\boldsymbol{\mu})$, Eq. (4), and the fact that $\langle g_i(\mathbf{x}) \rangle_{p(\mathbf{x})} = \langle g_i(\mathbf{x}) \rangle_{q(\mathbf{x}|\boldsymbol{\mu})}$, which comes from Eq. (14).

We are interested in finding the maximal normalized bias that is consistent with a given $\Delta S$. Rather than maximizing the normalized bias at fixed $\Delta S$, we take the complementary approach: For each possible bias, we find the minimal possible $\Delta S$. This gives us a relationship between bias and minimal $\Delta S$, which we can invert to obtain the maximal bias for a given $\Delta S$. Since $S_q(\boldsymbol{\mu})$ is independent of $p(\mathbf{x})$, minimizing $\Delta S$ is equivalent to maximizing $S_p$ (see Eq. (17)). Thus, again we have a maximum entropy problem. Now, though, we have an additional constraint on the normalized bias, which gives us an additional Lagrange multiplier in addition to the $\lambda_i$ we had for the original optimization problem. This leads to (in analogy to Eq. (4))

$$p(\mathbf{x}|\boldsymbol{\mu}, \beta) = \frac{\exp\left[\beta B(\mathbf{x}) + \sum_i \lambda_i(\boldsymbol{\mu}, \beta) g_i(\mathbf{x})\right]}{Z(\boldsymbol{\mu}, \beta)} \tag{18}$$

where $Z(\boldsymbol{\mu}, \beta)$ is the partition function and the $\lambda_i(\boldsymbol{\mu}, \beta)$ are chosen to satisfy Eq. (2), but with $p(\mathbf{x})$ replaced by $p(\mathbf{x}|\boldsymbol{\mu}, \beta)$. Amongst all models that satisfy the moments constraints and have the same normalized bias, this is the one that is closest (in KL–divergence) to the maximum entropy model.

Note that we have slightly abused notation: whereas in the previous sections the $\lambda_i$ and $Z$ depended only on $\boldsymbol{\mu}$, they now depend on both $\boldsymbol{\mu}$ and $\beta$. However, the previous variables are closely related to the new ones: when $\beta = 0$ the constraint associated with $b$ disappears, and we recover $q(\mathbf{x}|\mu)$; that is, $p(\mathbf{x}|\boldsymbol{\mu}, 0) = q(\mathbf{x}|\mu)$. Consequently, $\lambda_i(\boldsymbol{\mu}, 0) = \lambda_i(\boldsymbol{\mu})$, and $Z(\boldsymbol{\mu}, 0) = Z(\boldsymbol{\mu})$.

Relating $\Delta S$ to $b$ is now a purely numerical task: choose a set of $\mu_i$ and a normalized bias, $b$, determine the Lagrange multipliers, $\lambda_i(\boldsymbol{\mu}, \beta)$ and $\beta$, that appear in Eq. (18), then compute $S_p$ the entropy of $p(\mathbf{x}|\boldsymbol{\mu}, \beta)$, and subtract that from $S_q(\mu)$ to find $\Delta S$ (see Eq. (17)). In section 3.2 we do exactly that. First, however, to gain some intuition into how the normalized bias depends on $\Delta S$, we compute the relationship between the two perturbatively. This can be done by considering the small $\beta$ limit. In this limit we can expand both $\Delta S$ and $b$ as a Taylor series in $\beta$. Defining

$$\Delta S(\beta) \equiv S_q(\boldsymbol{\mu}) - S_p(\beta) \tag{19}$$

where $S_p(\beta)$ is the entropy of $p(\mathbf{x}|\boldsymbol{\mu}, \beta)$, and using primes to denote derivatives with respect to $\beta$, we have, through second order in $\beta$,

$$\Delta S(\beta) = S_q(\boldsymbol{\mu}) - S_p(0) - \beta S_p'(0) - \frac{\beta^2}{2} S_p''(0) \tag{20a}$$

$$b(\beta) = b(0) + \beta b'(0) \,. \tag{20b}$$

We expand $\Delta S(\beta)$ to second order in $\beta$ because $S'_p(0) = 0$, which follows from the fact that when $\beta \neq 0$ there is an additional constraint on the normalized bias, and so any $\beta \neq 0$ can only lower the entropy; therefore, $\beta = 0$ must be a local maximum. Alternatively, a straightforward calculation in which we write down the entropy of $p(\mathbf{x}|\boldsymbol{\mu}, \beta)$ using Eq. (18) (which results in an expression analogous to Eq. (6)) and differentiate with respect to $\beta$, yields

$$S'_p(\beta) = -\beta b'(\beta) \,. \tag{21}$$

From this it follows that $S'_p(0) = 0$; in addition, we see that $S''_p(0) = -b'(0)$. Thus, using the fact that when $\beta = 0$, $p(\mathbf{x}|\boldsymbol{\mu}, 0)$ is within the model class, so $S_p(0) = S_q(\boldsymbol{\mu})$, Eq. (20) tells us that when $\beta$ is sufficiently small,

$$\Delta S = \frac{(b - m)^2}{2b'(0)} \,. \tag{22}$$

The term in the denominator, $b'(0)$, is relatively easy to compute, and we show in Appendix C (in the supplementary material) that it is given by

$$b'(0) = \text{Var}[B]_{q(\mathbf{x}|\boldsymbol{\mu})} - \sum_{i,j=1}^{m} \langle B(x)\delta g_i(x)\rangle_{q(x|\boldsymbol{\mu})} C_{ij}^{q^{-1}} \langle \delta g_j(x)B(x)\rangle_{q(x|\boldsymbol{\mu})} \,. \tag{23}$$

The key result of the perturbative analysis is that when the true distribution is out of the model class, the normalized bias can be increased by a term proportional to $b'(0)^{1/2}$. Thus, the size of $b'(0)$ is crucial for telling us how big the bias really is. In the next section we investigate this numerically for a particular model, the Ising model.

## 3 Numerical Results: Estimation bias in Ising models

For our numerical simulations, we consider the second order maximum entropy model on $n$ binary variables, also known as the Ising model [12] (see [13, 14] for an application of Ising models to neuroscience). In this section, we use numerical studies to verify that the asymptotic bias gives an accurate characterization of the expected bias for relevant sample-sizes $K$, investigate the size of the normalized bias when the true model is not in the model class, and study the scaling of the normalized bias with the number of parameters. We show numerically that, for the Ising model, the model-misspecification can result in the normalized bias increasing rapidly with population size.

### 3.1 Estimation in a binary maximum entropy model

We consider $n$ interacting spins $s_i$, $i = 1, ..., n$ with $s_i \in \{0, 1\}$. We put constraints on the first and second moments only, so $m$, the number of constraints, is $n(n + 1)/2$: $g_i(\mathbf{s}) = s_i$ and $g_{ij}(\mathbf{s}) = s_i s_j$, $i < j$. The maximum entropy model (with the $\lambda_i$'s replaced by $h_i$ and $J_{ij}$ and the $g_i$ written explicitly) has the form

$$q(\mathbf{s}|h, J) = \frac{1}{Z(h, J)} \exp\left[\sum_i h_i s_i + \sum_{i<j} s_i J_{ij} s_j\right] \,. \tag{24}$$

To illustrate our results for the asymptotic bias, and to investigate how large $K$ has to be for the asymptotic calculation to be relevant, we performed the following simulations: For different values of $K$ (ranging from 10 to $10^4$) and different values of the model-size $n \in \{2, 3, 5, 10, 15\}$, we generated $10^4$ data sets of size $K$ each from an independent binary model with $n$ variables and mean $\mu = 0.1$ or $\mu = 0.5$, i.e. sampling from the distribution given in Eq. (24) with $J_{ij} = 0$ and $h_i = \log(\mu/(1 - \mu))$. For each such data set, we fit a pairwise binary maximum entropy model to the data by gradient-ascent on the (log-concave) likelihood. By calculating the entropy of the resulting model (via Eq. (6)) and averaging over the $10^4$ data sets, we obtained a numerical estimate of the difference between the true entropy and the expected estimated entropy; i.e. the bias.

Figure 1 shows (aside from the reassuring fact that our asymptotic calculations are consistent with the numerical simulations) that the asymptotic solution gives surprisingly accurate results even for relatively low values of $K$. From figures 1B and D, we can see that, for values of $K$ of around 100, the numerical biases already lie very close to the asymptotic prediction. Since the asymptotics are accurate for large $K$, we expect this fit to remain close. While we did observe some deviations

for very large data sets for which the bias is very small ($K > 10^3$), such deviations could be a consequence of numerical errors in the fitting-procedure.

We note that our choice of $J_{ij} = 0$ is merely for concreteness, and that the validity of our formulation is not dependent on the values of $J_{ij}$. We also performed simulations with models in which $J_{ij}$ is non-zero and drawn from a Gaussian distribution, which yielded qualitatively similar results.

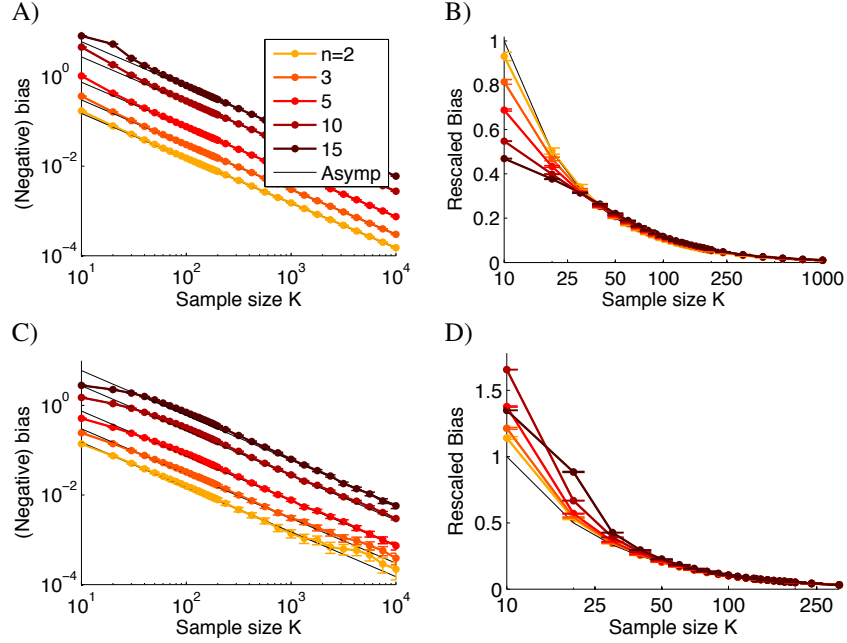

Figure 1: **Asymptotic bias in Ising models. A)** Comparison of asymptotic bias with expected bias calculated via simulations of an independent model with a mean of 0.5 (see text). The thin-black lines correspond to the bias as predicted by our asymptotic calculation. We have here inverted the sign of the bias, the actual biases are negative numbers. **B)** Same data as in A, but on a semi-log plot to illustrate how many samples are necessary for the asymptotic bias to be an accurate representation of the actual bias: For the parameters used here, the bias seems to be accurate even for small ($< 100$) values of $K$. We rescaled the estimated biases of each population size $n$ such that the predicted asymptotic biases (thin black lines) are on top of each other, and such that the biases are positive. **C** and **D)** Same as in A and B, but for an independent model with mean 0.1. Error bars show standard errors on the mean estimates from $10^4$ simulated data sets.

## 3.2   Estimation bias when the data has higher-order correlations

What happens when the true model is not in the model class? To investigate this question, we first consider homogeneous pairwise maximum entropy models ($h_i = h$ and $J_{ij} = J$) of sizes $n \in \{5, 10, 15\}$, common means $\langle s_i \rangle = 0.5$ or 0.1, and pairwise correlation-coefficient $\rho_{i,j} = 0.1$ for each pair $i, j$. For a range of normalized biases, we calculated $\Delta S$, the maximum entropy difference between $g(\mathbf{x}|\boldsymbol{\mu})$ and an out of model class distribution as a function of normalized bias, $b$. For very small or large normalized biases, the optimization did not converge to values moment constraints, indicating that such an extreme normalized bias would be inconsistent with the specified second order moments. The results are shown in Fig. 2, along with the perturbative predictions. For these choices of parameters, the maximum and minimum normalized bias did not deviate much from the within-model-class case. In the next example, we illustrate that the deviation can be very large.

To get a better understanding of the additional bias (or, potentially, reduction in bias) due to model misspecification, we studied the bias of the *Dichotomized Gaussian* distribution, which can be interpreted as a very simple model of neural population activity in which correlations among neurons are induced by common, Gaussian inputs into threshold neurons [21, 22]. In this case we simply set $p(\mathbf{x})$ to a Dichotomized Gaussian, and numerically computed the bias and the KL–divergence between $p(\mathbf{x})$ and the maximum entropy model with the same first and second moments. We did

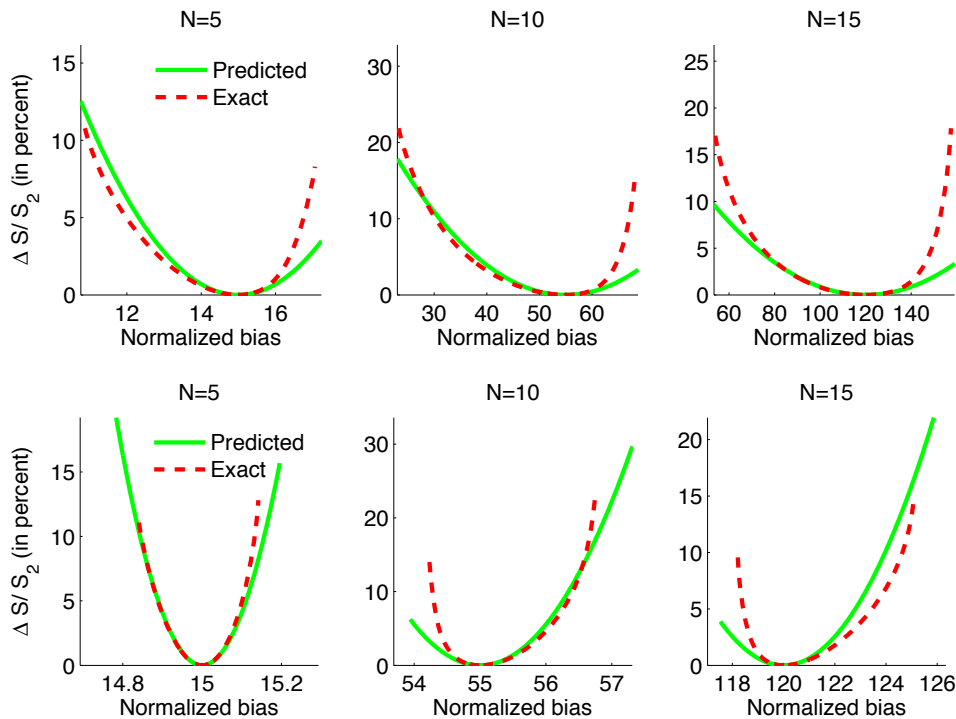

Figure 2: **Bias in the case of model misspecification. Top row:** $\Delta S/S_2$, where $S_2$ is the entropy of the second order model, as a function of the normalized bias for a model with means $\langle s_i \rangle = 0.5$ and correlation-coefficient $0.1$. The red (dashed) lines show the exact $\Delta S$ calculated by using equation (18), and the green (solid) lines using the perturbative expansion in equation (22). The curves end because for normalized biases too large or too small the optimization does not converge to values which satisfy the moment constraints. **Bottom row:** Same as top row, but using means of $\langle s_i \rangle = 0.1$.

this for means set to $\langle s_i \rangle = 0.02$, a realistic value for applications of maximum entropy models in neuroscience, and different values of the pairwise correlation coefficient $\rho \in \{0.02, 0.1, 0.5\}$. We also included, for comparison, the normalized bias for a within model class distribution (i.e. a maximum entropy model with matched first and second moments), which is just $n(n+1)/2$.

For the Dichotomized Gaussian, the normalized bias was substantially larger than the within model class bias. For example, for population size $n = 15$, its bias is 2.3 times larger for $\rho = 0.1$, and 6.8 times larger for $\rho = 0.5$. Figure 3B shows $\Delta S$ versus population size for the models in Fig. 3A, and the corresponding "maximally biased" model; i.e. the model which has the same normalized bias as the Dichotomized Gaussian, but minimal $\Delta S$. Interestingly, $\Delta S$ for the maximally biased models (equation (18)) is very similar to $\Delta S$ for the Dichotomized Gaussian. This suggests that our extremal calculation of the bias is relevant for a reasonably mechanistic model of neural population activity.

## 4    Conclusions

In recent years, there has been a resurgence of interest in maximum entropy models in neuroscience and related fields [13, 14, 15]. In particular, maximum entropy models can be useful for model-based estimation of the information content of neural populations [11], as direct information-estimates do not scale well for large population sizes. In this paper, we studied estimation biases in the entropy of maximum entropy models. We focused on "naive" estimators, i.e. estimators of the entropy which simply calculate it from the empirical estimates of the probabilities of the model, and do not attempt to do any bias reduction.

We found that if the true model is in the model class, the (downward) bias in a maximum entropy estimate from finite observations is proportional to the ratio of the number of parameters to the number of observations, a relationship which is identical to that of the (naive) histogram estimators [8, 9]. However, we also show that if the model is misspecified (i.e. if the true data do not come

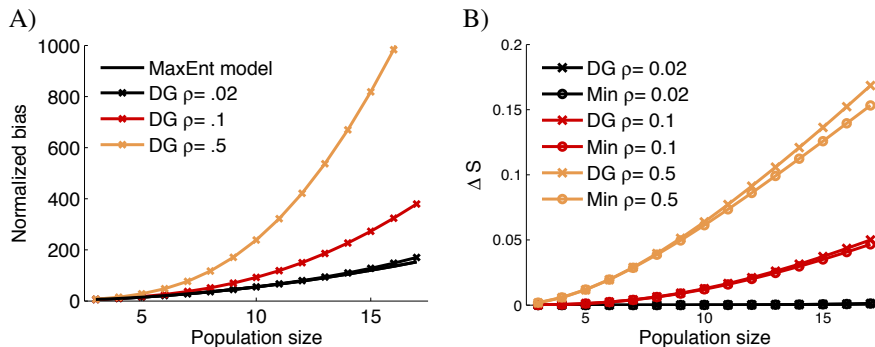

Figure 3: **Bias in the case of model misspecification, using the Dichotomized Gaussian. A)** Scaling of the normalized bias with population size. The normalized bias of the Dichotomized Gaussian (DG) is much larger than that of the maximum entropy model. **B)** Distance from model class, $\Delta S$, versus population size for the Dichotomized Gaussian and maximum entropy models. They are about the same, indicating that the Dichotomized Gaussian model has close to maximum bias.

from the specified exponential family model), then the bias can be much larger. We numerically investigated the bias in second-order binary maximum entropy models (also known as Ising models), and showed that in this case, model misspecification can lead to substantially bigger biases.

Non-parametric estimation of entropy is a well researched subject, and various estimators with optimized properties have been proposed (see e.g. [5, 23]). A number of studies have looked at the entropy estimation for the multivariate normal distribution [24, 25, 26, 27] and other continuous distributions, and improved estimators for the Gaussian distribution have been described [28]. As the (differential) entropy of a Gaussian distribution is essentially its log-determinant, the bias of this model can be related to results about the eigenvalues of random matrices [29]. An overview of estimators of the entropy of continuous-valued distributions is given in [30].

However, to our knowledge, the entropy bias of maximum entropy models in the presence of model-misspecification has not be characterized or studied numerically. We provided here an asymptotic derivation of this bias, and studied it numerically for the pairwise binary maximum entropy model, the Ising model. Our characterization of the bias relates the (worst case) bias in the case of model-misspecification to the distance (as measured by KL–divergence) between the model and the actual data. This characterization does not yield a precise estimate of the bias on a given data-set which could simply be 'subtracted-off'– thus, our derivation does not directly yield an improved estimator of the bias for such data-sets. However, importantly, our results show that model-misspecification can indeed lead to additional bias which can be much larger than generally appreciated. Using numerical simulations, we showed that this also happens for a realistic model which shares many properties with neural recordings. In addition, our results could be useful for deriving general guideline for how many samples a neurophysiological data-set needs to contain to achieve a bias which is less than some desired accuracy.

## Acknowledgements

We acknowledge support from the Gatsby Charitable Foundation. JHM is supported by an EC Marie Curie Fellowship, and IM in part by the IST Programme of the European Community, under the PASCAL2 Network of Excellence, IST-2007-216886. This publication only reflects the authors' views.

## References

[1] C.E. Shannon and W. Weaver. *The mathematical theory of communication*. University of Illinois Press, 1949.

[2] T.M. Cover, J.A. Thomas, J. Wiley, et al. *Elements of information theory*, volume 6. Wiley Online Library, 1991.

[3] F. Rieke, D. Warland, R. de R uytervansteveninck, and W. Bialek. *Spikes: exploring the neural code (computational neuroscience)*. The MIT Press, 1999.

[4] A. Borst and F. E. Theunissen. Information theory and neural coding. *Nat Neurosci*, 2(11):947–957, 1999 Nov.

[5] L. Paninski. Estimation of entropy and mutual information. *Neural Computation*, 15(6):1191–1253, 2003.

[6] B. B. Averbeck, P. E. Latham, and A. Pouget. Neural correlations, population coding and computation. *Nature Reviews Neuroscience*, 7(5):358–66, 2006.

[7] R. Quian Quiroga and S. Panzeri. Extracting information from neuronal populations: information theory and decoding approaches. *Nat Rev Neurosci*, 10(3):173–185, 2009.

[8] G. Miller. Note on the bias of information estimates. In *Information Theory in Psychology II-B*, chapter 95-100. Free Press, Glencole, IL, 1955.

[9] A. Treves and S. Panzeri. The upward bias in measures of information derived from limited data samples. *Neural Computation*, 7(2):399–407, 1995.

[10] S. Panzeri, R. Senatore, M. A. Montemurro, and R. S. Petersen. Correcting for the sampling bias problem in spike train information measures. *J Neurophysiol*, 98(3):1064–1072, 2007.

[11] Robin A A Ince, Alberto Mazzoni, Rasmus S Petersen, and Stefano Panzeri. Open source tools for the information theoretic analysis of neural data. *Front Neurosci*, 4, 2010.

[12] E. Ising. Beitrag zur Theorie des Ferromagnetismus. *Z. Phys*, 31:253, 1925.

[13] E. Schneidman, M. J. 2nd Berry, R. Segev, and W. Bialek. Weak pairwise correlations imply strongly correlated network states in a neural population. *Nature*, 440(7087):1007–12, 2006.

[14] J. Shlens, G. D. Field, J. L. Gauthier, M. I. Grivich, D. Petrusca, A. Sher, A. M. Litke, and E. J. Chichilnisky. The structure of multi-neuron firing patterns in primate retina. *J Neurosci*, 26(32):8254–66, 2006.

[15] I. E. Ohiorhenuan, F. Mechler, K. P. Purpura, A. M. Schmid, Q. Hu, and J. D. Victor. Sparse coding and high-order correlations in fine-scale cortical networks. *Nature*, 466(7306):617–621, 2010.

[16] G. Tkacik, E. Schneidman, M. J. Berry, II, and W. Bialek. Spin glass models for a network of real neurons. *arXiv:q-bio/0611072v2*, 2009.

[17] Y. Roudi, J. Tyrcha, and J. Hertz. Ising model for neural data: model quality and approximate methods for extracting functional connectivity. *Phys Rev E Stat Nonlin Soft Matter Phys*, 79(5 Pt 1):051915, May 2009.

[18] Y. Roudi, E. Aurell, and J. A. Hertz. Statistical physics of pairwise probability models. *Front Comput Neurosci*, 3:22, 2009.

[19] T. Mora, A. M. Walczak, W. Bialek, and C. G. Jr Callan. Maximum entropy models for antibody diversity. *Proc Natl Acad Sci U S A*, 107(12):5405–5410, 2010.

[20] A.W. Van der Vaart. *Asymptotic statistics*. Cambridge Univ Pr, 2000.

[21] J.H. Macke, P. Berens, A.S. Ecker, A.S. Tolias, and M. Bethge. Generating spike trains with specified correlation coefficients. *Neural Computation*, 21(2):397–423, 2009.

[22] J.H. Macke, M. Opper, and M. Bethge. Common input explains higher-order correlations and entropy in a simple model of neural population activity. *Physical Review Letters*, 106(20):208102, 2011.

[23] I. Nemenman, W. Bialek, and R.D.R. Van Steveninck. Entropy and information in neural spike trains: Progress on the sampling problem. *Physical Review E*, 69(5):056111, 2004.

[24] N.A. Ahmed and D. V. Gokhale. Entropy expressions and their estimators for multivariate distributions. *Information Theory, IEEE Transactions on*, 35(3):688–692, 1989.

[25] O. Oyman, R. U. Nabar, H. Bolcskei, and A. J. Paulraj. Characterizing the statistical properties of mutual information in MIMO channels: insights into diversity-multiplexing tradeoff. In *Signals, Systems and Computers, 2002. Conference Record of the Thirty-Sixth Asilomar Conference on*, volume 1, pages 521–525. IEEE, 2002.

[26] N. Misra, H. Singh, and E. Demchuk. Estimation of the entropy of a multivariate normal distribution. *Journal of multivariate analysis*, 92(2):324–342, 2005.

[27] G. Marrelec and H. Benali. Large-sample asymptotic approximations for the sampling and posterior distributions of differential entropy for multivariate normal distributions. *Entropy*, 13(4):805–819, 2011.

[28] S. Srivastava and M.R. Gupta. Bayesian estimation of the entropy of the multivariate Gaussian. In *Information Theory, 2008. ISIT 2008. IEEE International Symposium on*, pages 1103–1107. IEEE, 2008.

[29] N.R. Goodman. The distribution of the determinant of a complex Wishart distributed matrix. *The Annals of mathematical statistics*, 34(1):178–180, 1963.

[30] M. Gupta and S. Srivastava. Parametric Bayesian estimation of differential entropy and relative entropy. *Entropy*, 12(4):818–843, 2010.

